# On Learning $\mu$-Perceptron Networks with Binary Weights

**Mostefa Golea**
Ottawa-Carleton Institute for Physics
University of Ottawa
Ottawa, Ont., Canada K1N 6N5
050287@acadvm1.uottawa.ca

**Mario Marchand**
Ottawa-Carleton Institute for Physics
University of Ottawa
Ottawa, Ont., Canada K1N 6N5
mmmsj@acadvm1.uottawa.ca

**Thomas R. Hancock**
Siemens Corporate Research
755 College Road East
Princeton, NJ 08540
hancock@learning.siemens.com

## Abstract

Neural networks with binary weights are very important from both the theoretical and practical points of view. In this paper, we investigate the learnability of single binary perceptrons and unions of $\mu$-binary-perceptron networks, i.e. an "OR" of binary perceptrons where each input unit is connected to one and only one perceptron. We give a polynomial time algorithm that PAC learns these networks under the uniform distribution. The algorithm is able to identify both the network connectivity and the weight values necessary to represent the target function. These results suggest that, under reasonable distributions, $\mu$-perceptron networks may be easier to learn than fully connected networks.

## 1   Introduction

The study of neural networks with binary weights is well motivated from both the theoretical and practical points of view. Although the number of possible states in the weight space of a binary network are finite, the capacity of the network is not much inferior to that of its continuous counterpart (Barkai and Kanter 1991). Likewise, the hardware realization of binary networks may prove simpler.

A major obstacle impeding the development of binary networks is that, under an arbitrary distribution of examples, the problem of learning binary weights is NP-complete(Pitt and Valiant 1988). However, the question of the learnability of this class of networks under some *reasonable* distributions is still open. A first step in this direction was reported by Venkatesh (1991) for single majority functions.

In this paper we investigate, within the PAC model (Valiant 1984; Blumer *et al.* 1989), the learnability of two interesting concepts under the uniform distribution: 1) Single binary perceptrons with arbitrary thresholds and 2) Unions of $\mu$-binary perceptrons, *i.e.* an "OR" of binary perceptrons where each input unit is connected to one and only one perceptron (fig. 1) [1]. These functions are related to so-called $\mu$ or read-once formulas (Kearns *et al.* 1987). Learning these functions on special distributions is presently a very active research area (Pagallo and Haussler 1989; Valiant and Warmuth 1991). The $\mu$ restriction may seem to be a severe one but it is not. First, under an arbitrary distribution, $\mu$-formulas are not easier to learn than their unrestricted counterpart (Kearns *et al.* 1987). Second, the $\mu$ assumption brings up a problem of its own: determining the network *connectivity*, which is a challenging task in itself.

Our main results are polynomial time algorithms that PAC learn single binary perceptrons and unions of $\mu$-binary perceptrons under the uniform distribution of examples. These results suggest that $\mu$-perceptron networks may be somewhat easier to learn than fully connected networks if one restricts his attention to reasonable distributions of examples. Because of the limited space, only a sketch of the proofs is given in this abstract.

## 2    Definitions

Let $I$ denote the set $\{0, 1\}$. A perceptron $g$ on $I^n$ is specified by a vector of $n$ real valued weights $w_i$ and a single real valued threshold $\theta$. For $\mathbf{x} = (x_1, x_2, ..., x_n) \in I^n$, we have:

$$g(\mathbf{x}) = \begin{cases} 1 & \text{if } \sum_{i=1}^{i=n} w_i x_i \geq \theta \\ 0 & \text{if } \sum_{i=1}^{i=n} w_i x_i < \theta \end{cases} \tag{1}$$

A perceptron is said to be positive if $w_i \geq 0$ for $i = 1, ..., n$. We are interested in the case where the weights are binary valued ($\pm 1$). We assume, without loss of generality (w.l.o.g.), that $\theta$ is integer.

A function $f$ is said to be a union of perceptrons if it can be written as a disjunction of perceptrons. If these perceptrons do not share any variables, $f$ is said to be a union of $\mu$-perceptrons (fig. 1), and we write

$$f = g^{(1)} \vee g^{(2)} \vee ... \vee g^{(s)} \quad 1 \leq s \leq n \tag{2}$$

We shall assume, w.l.o.g., that $f$ is expressed with the maximum possible number of $g^{(i)}$'s, and has the minimum possible number of non-zero weights.

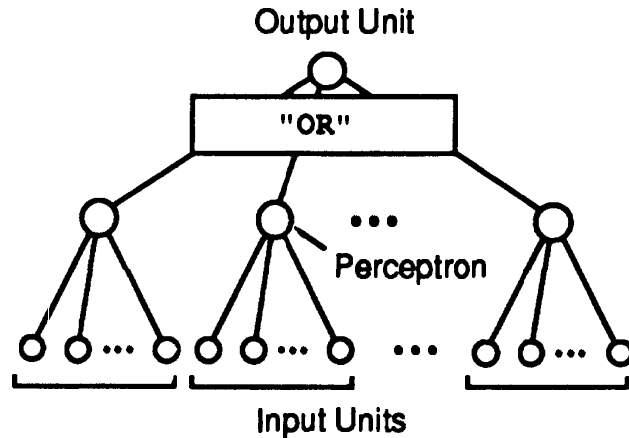

**Output Unit**

**Input Units**

Figure 1: A two layer network representing a union of $\mu$-perceptrons. Note that each input unit is connected to one and only one perceptron (hidden unit). The output unit computes an OR function.

We denote by $P(A)$ the probability of an event $A$ and by $\hat{P}(A)$ its empirical estimate based on a given finite sample. All probabilities are taken with respect to the uniform distribution $D$ on $I^n$. If $a \in \{0,1\}$, we denote by $P(f = 1|x_i = a)$ the conditional probability that $f = 1$ given the fact that $x_i = a$.

The algorithm will make use of the following probabilistic quantities:

The *influence* of a variable $x_i$, denoted $Inf(x_i)$, is defined as

$$Inf(x_i) = P(f = 1|x_i = 1) - P(f = 1|x_i = 0).$$

Intuitively, the influence of a variable is positive (negative) if its weight is positive (negative).

The *correlation* of a variable $x_j$ with $x_i$, denoted $C(x_i, x_j)$, is defined as

$$C(x_i, x_j) = \frac{P(f = 1|x_i x_j = 1) - P(f = 1|x_i = 1)}{P(f = 1|x_j = 1) - P(f = 1)}$$

where $x_i x_j = 1$ stands for $x_i = x_j = 1$. This quantity depends largely on whether or not the two variables are in same perceptron.

In what follows, we adopt the PAC learning model (Valiant 1984; Blumer *et al.* 1989). Here the methodology is to draw a sample of a certain size labeled according to the unknown target function $f$ and then to find a "good" approximation $h$ of $f$. The error of the hypothesis function $h$, with respect to the target $f$, is defined to be $P(h \neq f) = P(h(\mathbf{x}) \neq f(\mathbf{x}))$, where $\mathbf{x}$ is distributed according to $D$. An algorithm learns from examples a target class $F$ using an hypothesis class $H$ under the distribution $D$ on $I^n$, if for every $f \in F$, and any $0 < \epsilon, \delta < 1$, the algorithm runs in time polynomial in $(n, \epsilon, \delta)$ and outputs an hypothesis $h \in H$ such that

$$P[\ P(h \neq f) > \epsilon\ ] < \delta$$

## 3   Learning Networks with Binary Weights

### 3.1   Learning Single Binary Perceptrons

Let us assume that the target function $f$ is a single binary perceptron $g$ given by eq. (1). Let us assume also that the distribution generating the examples is uniform on $\{0,1\}^n$. The learning algorithm proceeds in two steps:

1. Estimating the weight values (signs). This is done by estimating the *influence* of each variable. Then the target perceptron is reduced to a *positive* perceptron by simply changing $x_i$ to $1 - x_i$ whenever $w_i = -1$.

2. Estimating the threshold of the positive target perceptron and hence the threshold of the original perceptron.

To simplify the analysis, we introduce the following notation. Let $N$ be the number of negative weights in $g$ and let $\mathbf{y}$ be defined as

$$y_i = \begin{cases} x_i & \text{if } w_i = 1 \\ 1 - x_i & \text{if } w_i = -1 \end{cases} \tag{3}$$

Then eq. (1) can be written as

$$g(\mathbf{y}) = \begin{cases} 1 & \text{if } \sum_{i=1}^{i=n} y_i \geq \Omega \\ 0 & \text{if } \sum_{i=1}^{i=n} y_i < \Omega \end{cases} \tag{4}$$

where the renormalized threshold $\Omega$ is related to the original threshold $\theta$ by: $\Omega = \theta + N$. We assume, w.l.o.g., that $1 \leq \Omega \leq n$. Note that $D(\mathbf{x}) = D(\mathbf{y})$.

The following lemma, which follows directly from Bahadur's expansion (Bahadur 1960), will be used throughout this paper to approximate binomial distributions.

**Lemma 1** *Let $d$ and $n$ be two integers. Then, if $\frac{n}{2} \leq d \leq n$,*

$$\sum_{i=d}^{n} \binom{n}{i} \leq \frac{1}{2} \binom{n}{d} \frac{1}{1-z}$$

*where $z = \frac{1}{2} \frac{n+1}{d+1}$. And if $0 \leq d \leq \frac{n}{2}$,*

$$\sum_{i=0}^{d} \binom{n}{i} \leq \frac{1}{2} \binom{n}{d} \frac{1}{1-z'}$$

*where $z' = \frac{1}{2} \frac{n+1}{n-d+1}$.*

As we said earlier, intuition suggests that the influence of a variable is positive (negative) if its weight is positive (negative). The following lemma strengthens this intuition by showing that there is a measurable *gap* between the two cases. This gap will be used to estimate the weight values (signs).

**Lemma 2** *Let $g$ be a perceptron such that $P(g = 1)$, $P(g = 0) > \rho$, where $0 < \rho < 1$. Then,*

$$Inf(x_i) \begin{cases} > \frac{\rho}{n+2} & \text{if } w_i = 1 \\ < -\frac{\rho}{n+2} & \text{if } w_i = -1 \end{cases}$$

**Proof sketch for $w_i = 1$:** We exploit the equivalence between eq. (1) and eq. (4), and the fact that the distribution is uniform:

$$
\begin{aligned}
Inf(x_i) &= P(g(\mathbf{x}) = 1|x_i = 1) - P(g(\mathbf{x}) = 1|x_i = 0) \\
&= P(g(\mathbf{y}) = 1|y_i = 1) - P(g(\mathbf{y}) = 1|y_i = 0) \\
&= 2\frac{\dbinom{n-1}{\Omega-1}}{\sum_{i=\Omega}^{n}\dbinom{n}{i}}P(g = 1) \qquad (5)\\
&= 2\frac{\dbinom{n-1}{\Omega-1}}{\sum_{i=0}^{\Omega-1}\dbinom{n}{i}}P(g = 0) \qquad (6)
\end{aligned}
$$

Applying lemma 1 to either eq. (5) (case: $\Omega \geq \frac{n}{2}$) or eq.( 6) (case: $\Omega \leq \frac{n}{2}$) yields the desired result.□

Once we determine the weight values, we reduce $g$ to its positive form by changing $x_i$ to $1 - x_i$ whenever $w_i = -1$. The next step is to estimate the renormalized threshold $\Omega$ and hence, the original threshold $\theta$.

**Lemma 3** *Let $g$ be the positive perceptron with a renormalized threshold $\Omega$. Let $g'$ the positive perceptron obtained from $g$ by substituting $r$ for $\Omega$. Then, if $r \leq \Omega$,*

$$P(g \neq g') \leq 1 - P(g = 1|g' = 1)$$

So, if we estimate $P(g = 1|g' = 1)$ for $r = 1, 2, \ldots$ and then choose as the renormalized threshold the *least* $r$ for which $P(g = 1|g' = 1) \geq (1 - \epsilon)$, we are guaranteed to have $P(g \neq g') \leq \epsilon$. Obviously, such $r$ exists and is always $\leq \Omega$ because $P(g = 1|g' = 1) = 1$ for $r = \Omega$.

A sketch of the algorithm for learning single binary perceptrons is given in fig. 2.

**Theorem 1** *The class of binary perceptrons is PAC learnable under the uniform distribution.*

**Proof sketch:** A sample of size $O(\frac{n^2}{\epsilon^4} \ln \frac{n}{\delta})$ is sufficient to ensure that the different probabilities are estimated to within a sufficient precision (Hagerup and Rub 1989). Steps 2 and 3 of the algorithm are obvious. If we reach step 5, $P(g = 1) > \frac{\epsilon}{2}$ and $P(g = 0) > \frac{\epsilon}{2}$. Then one has only to set $\rho = \frac{\epsilon}{2}$ and apply lemma 2 and 3 to step 5 and 6 respectively. Finally, the algorithm runs in time polynomial in $n$, $\epsilon$ and $\delta$.□

## 3.2 Learning Unions of μ-binary Perceptrons

Let us assume now that the target function $f$ is a union of $\mu$-binary perceptrons as in eq. (2). Note that we do not assume that the architecture (connectivity) is known in advance. Rather, it is the task of the learning algorithm to determine which variables are in a given perceptron. The learning algorithm proceeds in three steps:

---

**Algorithm** *LEARN-BINARY-PERCEPTRON(n,ε,δ)*

**Parameters:** $n$ is the number of input variables, $\epsilon$ is the accuracy parameter and $\delta$ is the confidence parameter.

**Output:** a binary perceptron $h$.

**Description:**

1. Call $M = \frac{[80(n+2)]^2}{\epsilon^4} \ln \frac{8n}{\delta}$ examples. This sample is be used to estimate the different probabilities. Initialize $h$ to the constant perceptron 0. Initialize $N$ to 0.

2. (Are most examples positive?) If $\hat{P}(g = 1) \geq (1 - \frac{\epsilon}{4})$ then set $h = 1$. Return $h$.

3. (Are most examples negative?) If $\hat{P}(g = 1) \leq \frac{\epsilon}{4}$ then return $h$.

4. Set $\rho = \frac{\epsilon}{2}$.

5. (Reduce $g$ to a positive perceptron) For each input variable $x_i$:

   (a) If $\hat{Inf}(x_i) > \frac{1}{2}\frac{\rho}{n+2}$, set $w_i = 1$.

   (b) Else if $\hat{Inf}(x_i) < -\frac{1}{2}\frac{\rho}{n+2}$, set $N = N + 1$, $w_i = -1$, and change $x_i$ to $1 - x_i$.

   (c) Else delete $x_i$ (update $n$ accordingly).

6. (Estimating the bias) Let $g'$ be as defined in lemma 3. Initialize $r$ to 1.

   (a) Estimate $P(g = 1|g' = 1)$.

   (b) If $\hat{P}(g = 1|g' = 1) > 1 - \frac{1}{2}\epsilon$, set $\Omega = r$ and go to step 7.

   (c) $r = r + 1$. Go to step 6a.

7. Set $\theta = \Omega - N$. Return $h$ (that is $(w_1, ..., w_n; \theta)$).

---

Figure 2: An algorithm for learning single binary perceptrons.

1. Estimating the weight values (signs). This is again done by estimating the *influence* of each variable. Then the target function is reduced to a union of *positive* perceptrons by simply changing $x_i$ to $1 - x_i$ whenever $w_i = -1$.

2. Estimating which variables belong to the *same* perceptron. This is done by estimating correlations between different variables.

3. Estimating the renormalized threshold of the each positive target perceptron and hence, the threshold of the original perceptron.

The following lemma is a straightforward generalization of lemma 2.

**Lemma 4** *Let $f$ be a union of $\mu$-perceptrons as in (2). Let $g^{(a)}$ be a perceptron in $f$ and let $x_i \in g^{(a)}$. Assume that $P(f = 1) < 1 - \gamma$ and $P(g = 1), P(g = 0) > \rho$ where $0 < \gamma, \rho < 1$. Then*

$$Inf(x_i) \begin{cases} > & \frac{\gamma\rho}{n+2} & \text{if } w_i = 1 \\ < & -\frac{\gamma\rho}{n+2} & \text{if } w_i = -1 \end{cases}$$

**Proof sketch for $w_i = 1$:** Let $m$ be the disjunction of all perceptrons in $f$ except $g$. Then, using the inclusion-exclusion property and the fact that perceptrons in $f$ do not share any variables,

$$Inf(x_i) = (1 - P(m = 1))(P(g = 1|x_i = 1) - (P(g = 1|x_i = 0))$$

$$> \quad \gamma(P(g = 1|x_i = 1) - (P(g = 1|x_i = 0))) \quad > \quad \frac{\gamma\rho}{n+2} \qquad (7)$$

Inequality (7) follows from the fact that $1 - P(m = 1) > 1 - P(f = 1) > \gamma$ and from lemma 2.□

Lemma 4 enables us to determine the weight values and reduce $f$ to its positive form. Note that we can assume, w.l.o.g., that $\gamma \geq \epsilon/2$ and $\rho \geq \epsilon/2n$. The next step is to determine which variables belong to the same perceptron. Starting with a variable, say $x_i$, the procedure uses the correlation measure to decide whether or not another variable, say $x_j$, belongs to $x_i$'s perceptron. We appeal to the following lemma where we assume that $f$ is already reduced to its positive form.

**Lemma 5** *Let $f$ be a union of $\mu$-perceptrons (positive form). Let $g^{(a)}$ be a perceptron in $f$. Let $x_i \in g^{(a)}$ and let $x_j$ and $x_k$ be two other influential variables in $f$. Then*

$$C(x_i, x_j) - C(x_i, x_k) = \begin{cases} 0 & \text{if } x_j \in g^{(a)} \text{ and } x_k \in g^{(a)} \\ 0 & \text{if } x_j \notin g^{(a)} \text{ and } x_k \notin g^{(a)} \\ \geq \frac{1}{n^2} & \text{if } x_j \in g^{(a)} \text{ and } x_k \notin g^{(a)} \end{cases}$$

The lengthy proof of this important lemma will appear in the full paper.

If we estimate the correlations to within a sufficient precision, the *correlation gap*, established in lemma 5, enables us to decide which variables are in the same perceptron.

The last step is to estimate the bias of each perceptron. Let $g$ be a perceptron in $f$ and let $g'$ be the perceptron obtained from $g$ by setting its renormalized bias to $r$. Estimating $g$'s bias may be done by simply estimating $P(f = 1|g' = 1)$ for different values of the renormalized threshold, $r = 2, 3, ...$, and choosing the least $r$ such that $P(f = 1|g' = 1) \geq (1 - \epsilon/n)$ (see lemma 3 and step 6 in figure 2).

**Theorem 2** *The class of $\mu$-binary perceptron unions are PAC learnable under the uniform distribution.*

## 4   Conclusion and Open Problems

We presented a simple algorithm for PAC learning single binary perceptrons and $\mu$-binary-perceptron unions, under the uniform distribution of examples. The hardness results reported in the literature (see (Lin and Vitter 1991)) suggest that one can not avoid the training difficulties simply by considering only very simple neural networks. Our results (see also (Marchand and Golea 1992)) suggest that the combination of simple networks and reasonable distributions may be needed to achieve any degree of success.

The results reported here are part of an ongoing research aimed at understanding *nonoverlapping* perceptron networks (Hancock *et al.* 1991). The extension of these results to more complicated networks will appear in the full paper.

## Acknowledgements

M. Golea and M. Marchand are supported by NSERC grant OGP0122405. This research was conducted while T. Hancock was a graduate student at Harvard University, supported by ONR grant N00014-85-K-0445 and NSF grant NSF-CCR-89-02500.

## Footnotes

[1]The intersection is simply the complement of the union and can be treated similarly.

## References

[1] Bahadur R. (1960) "Some Approximations to the Binomial Distribution Function", *Annals Math. Stat.*, Vol.31, 43–54.

[2] Barkai E. & Kanter I. (1991) "Storage Capacity of a Multilayer Neural Network with Binary weights", *Europhys. Lett.*, Vol. 14, 107–112.

[3] Blumer A., Ehrenfeucht A., Haussler D., and Warmuth K. (1989) "Learnability and the Vapnik-Chervonenkis Dimension", *J. ACM*, Vol. 36, 929–965.

[4] Hagerup T. & Rub C. (1989) "A Guided Tour to Chernoff Bounds", *Info. Proc. Lett.*, Vol. 33, 305–308.

[5] Hancock T., Golea M., and Marchand M. (1991) "Learning Nonoverlapping Perceptron Networks From Examples and Membership Queries", TR-26-91, Center for Research in Computing Technology, Harvard University. *Submitted* to Machine Learning.

[6] Kearns M., Li M., Pitt L., and Valiant L. (1987) "On the Learnability of Boolean Formulas", in *Proc. of the 19th Annual ACM Symposium on Theory of Computing*, 285-295, New York, NY.

[7] Lin J.H. & Vitter J.S. (1991) "Complexity Results on Learning by Neural Nets", *Machine Learning*, Vol. 6, 211-230.

[8] Marchand M. & Golea M. (1992) "On Learning Simple Neural Concepts", to appear in *Network*.

[9] Pagallo G. & Haussler D. (1989) "A Greedy Method for learning $\mu$DNF functions under the Uniform Distribution". Technical Report UCSC-CRL-89-12, Santa Cruz: Dept. of Computer and Information Science, University of California at Santa Cruz.

[10] Pitt L. & Valiant L.G. (1988) "Computational Limitations on Learning from Examples", *J. ACM*, Vol. 35, 965-984.

[11] Valiant L.G. (1984) "A Theory of the Learnable", *Comm. ACM*, Vol. 27, 1134-1142.

[12] Valiant L.G. & Warmuth K. (Editors) (1991) *Proc. of the 4st Workshop on Computational Learning Theory*, Morgan Kaufman.

[13] Venkatesh S. (1991) "On Learning Binary Weights for Majority Functions", in *Proc. of the 4th Workshop on Computational Learning Theory*, 257–266, Morgan Kaufman.